# Hierarchical Transformation of Space in the Visual System

Alexandre Pouget          Stephen A. Fisher

Terrence J. Sejnowski
Computational Neurobiology Laboratory
The Salk Institute
La Jolla, CA 92037

## Abstract

Neurons encoding simple visual features in area V1 such as orientation, direction of motion and color are organized in retinotopic maps. However, recent physiological experiments have shown that the responses of many neurons in V1 and other cortical areas are modulated by the direction of gaze. We have developed a neural network model of the visual cortex to explore the hypothesis that visual features are encoded in head-centered coordinates at early stages of visual processing. New experiments are suggested for testing this hypothesis using electrical stimulations and psychophysical observations.

## 1  Introduction

Early visual processing in cortical areas V1, V2 and MT appear to encode visual features in eye-centered coordinates. This is based primarily on anatomical data and recordings from neurons in these areas, which are arranged in retinotopic maps. In addition, when neurons in the visual cortex are electrically stimulated [9], the direction of the evoked eye movement depends only on the retinotopic position of the stimulation site, as shown in figure 1. Thus, when a position corresponding to the left part of the visual field is stimulated, the eyes move toward the left (left figure), and eye movements in the opposite direction are induced if neurons on the right side are stimulated (right figure).

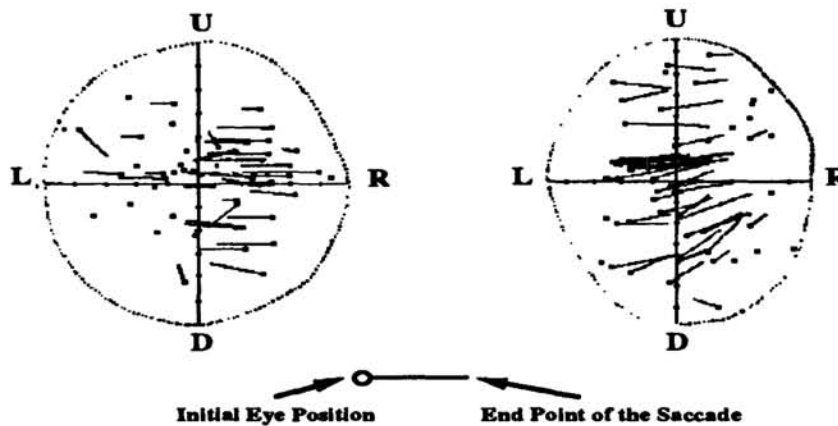

Figure 1: Eye Movements Evoked by Electrical Stimulations in V1

A variety of psychophysical experiments provide further evidence that simple visual features are organized according to retinal coordinates rather than spatiotopic coordinates [10, 5].

At later stages of visual processing the receptive fields of neurons become very large and in the posterior parietal cortex, containing areas believed to be important for sensory-motor coordination (LIP, VIP and 7a), the visual responses of neurons are modulated by both eye and head position [1, 2]. A previous model of the parietal cortex showed that the gain fields of the neurons observed there are consistent with a distributed spatial transformation from retinal to head-centered coordinates [14].

Recently, several investigators have found that static eye position also modulates the visual response of many neurons at early stages of visual processing, including the LGN, V1 and V3a [3, 6, 13, 12]. Furthermore, the modulation appears to be qualitatively similar to that previously reported in the parietal cortex and could contribute to those responses. These new findings suggest that coordinate transformations from retinal to spatial representations could be initiated much earlier than previously thought.

We have used network optimization techniques to study the spatial transformations in a feedforward hierarchy of cortical maps. The goals of the model were 1) to determine whether the modulation of neural responses with eye position as observed in V1 or V3a is sufficient to provide a head-centered coordinate frame, 2) to help interpret data based on the electrical stimulation of early visual areas, and 3) to provide a framework for designing experiments and testing predictions.

## 2   Methods

### 2.1   Network Task

The task of the network was to compute the head-centered coordinates of objects. If $\vec{E}$ is the eye position vector and $\vec{R}$ is the vector for the retinal position of the

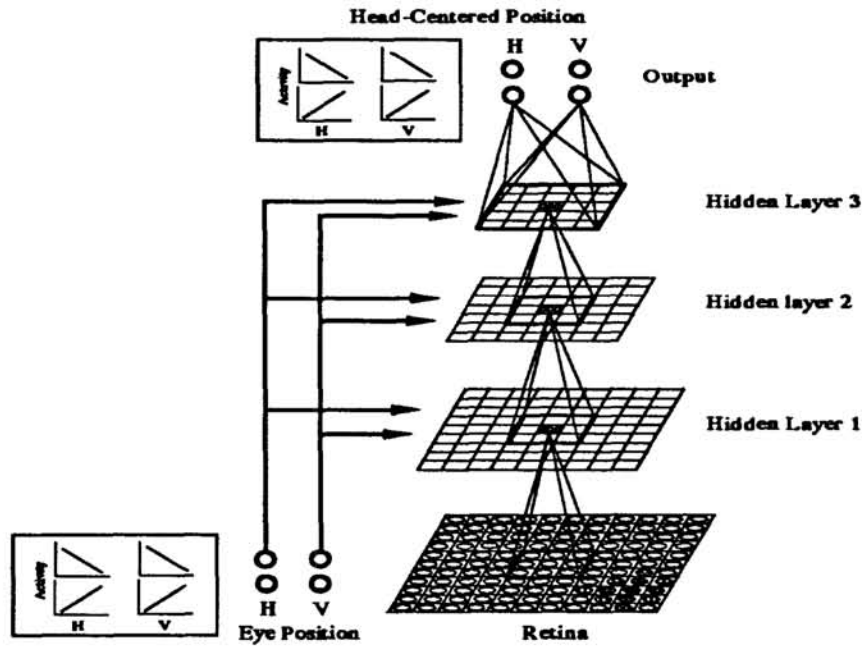

Figure 2: Network Architecture

object, then the head-centered position $\vec{P}$ is given by:

$$\vec{P} = \vec{R} + \vec{E} \tag{1}$$

A two layer network with linear units can solve this problem. However, the goal of our study was not to find the optimal architecture for this task, but to explore the types of intermediate representation developed in a multilayer network of non-linear units and to compare these results with physiological recordings.

## 2.2  Network Architecture

We trained a partially-connected multilayer network to compute the head-centered position of objects from retinal and eye position signals available at the input layer. Weights were shared within each hidden layer [7] and adjusted with the backprop-agation algorithm [11]. All simulations were performed with the SN2 simulator developed by Botou and LeCun.

In the hierarchical architecture illustrated in figure 2, the sizes of the receptive fields were restricted in each layer and several hidden units were dedicated to each location, typically 3 to 5 units, depending on the layer. Although weights were shared between locations within a layer, each type of hidden unit was allowed to develop its own receptive field properties. This architecture preserves two essential aspects of the visual cortex: 1) restricted receptive fields organized in retinotopic maps and 2) the sizes of the receptive fields increase with distance from the retina.

Training examples consisted of an eye position vector and a gaussian pattern of activity placed at a particular location on the input layer and these were system-

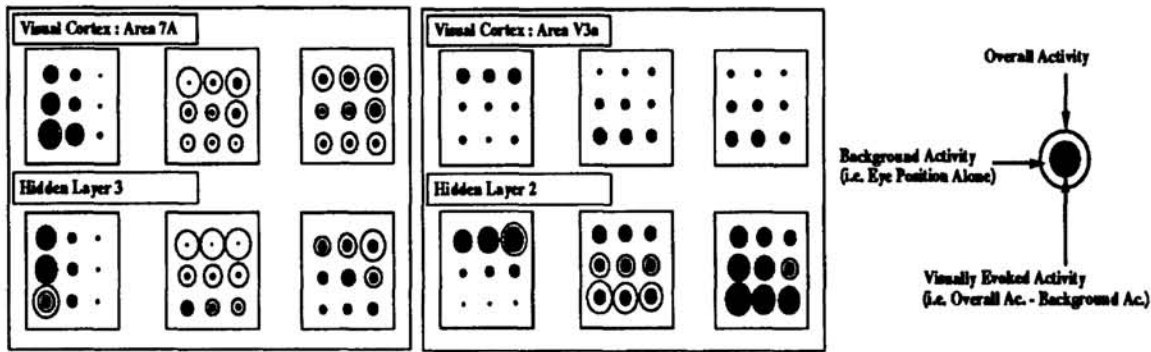

Figure 3: Spatial Gain Fields: Comparison Between Hidden Units and Cortical Neurons (background activity not shown for V3a neurons)

atically varied throughout the training. For some trials there were no visual inputs and the output layer was trained to reproduce the eye position.

### 2.3 Electrical Stimulation Experiments

Determining the head-centered position of an object is equivalent to computing the position of the eye required to foveate the object (i.e. for a foveated object $\vec{R} = 0$, which, according to equation 1, implies that $\vec{P} = \vec{E}$). Thus, the output of our network can be interpreted as the eye position for an intended saccadic eye movement to acquire the object.

For the electrical stimulation experiments we followed the protocol suggested by Goodman and Andersen [4] in an earlier study of the Zipser-Andersen model of parietal cortex [14]. The cortical model was stimulated by clamping the activity of a set of hidden units at a location in one of the layers to 1, their maximum values, and setting all visual inputs to 0. The changes in the activity of the output units were computed and interpreted as an intended saccade.

## 3   Results

We trained several networks with various numbers of hidden units per layer and found that they all converged to a nearly perfect solution in a few thousand sweeps through the training set.

### 3.1 Comparison Between Hidden Units and Cortical Neurons

The influence of eye position on the visual response of a cortical neuron is typically assessed by finding the visual stimulus eliciting the best response and measuring the gain of this response at nine different eye fixations [1]. The responses are plotted as circles with diameters proportional to activity and the set of nine circles is called the spatial gain field of a unit. We adopted the same procedure for studying the hidden units in the model.

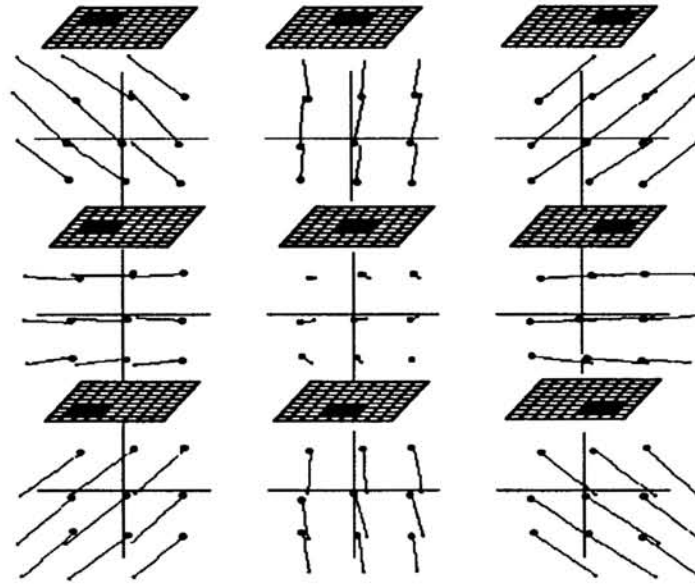

Figure 4: Eye Movements Evoked by Stimulating the Retina

The units in a fully-developed network have properties that are similar to those observed in cortical neurons (figure 3). Despite having restricted receptive fields, the overall activity of most units increased monotonically in one direction of eye position, each unit having a different preferred direction in head-centered space. Also, the inner and outer circles, corresponding to the visual activity and the overall activity (visual plus background) did not always increase along the same direction due to the nonlinear sigmoid squashing function of the unit.

## 3.2    Significance of the Spatial Gains Fields

Each hidden layer of the network has a retinotopic map but also contains spatiotopic (i.e. head-centered) information through the spatial gain fields. We call these retinospatiotopic maps.

At each location on a map, $\vec{R}$ is implicitly encoded by the position of a unit on the map, and $\vec{E}$ is provided by the inputs from the eye position units. Thus, each location contains all the information needed to recover $\vec{P}$, the head-centered coordinate. Therefore, all of the visual features in the map, such as orientation or color, are encoded in head-centered coordinates. This suggests that some visual representations in V1 and V3a may be retinospatiotopic.

## 3.3    Electrical Stimulation Experiments

Can electrical stimulation experiments distinguish between a purely retinotopic map, like the retina, and retinospatiotopic maps, like each of the hidden layers?

When input units in the retina are stimulated, the direction of the evoked movement is determined by the location of the stimulation site on the map (figure 4), as expected from a purely retinotopic map. For example, stimulating units in the upper

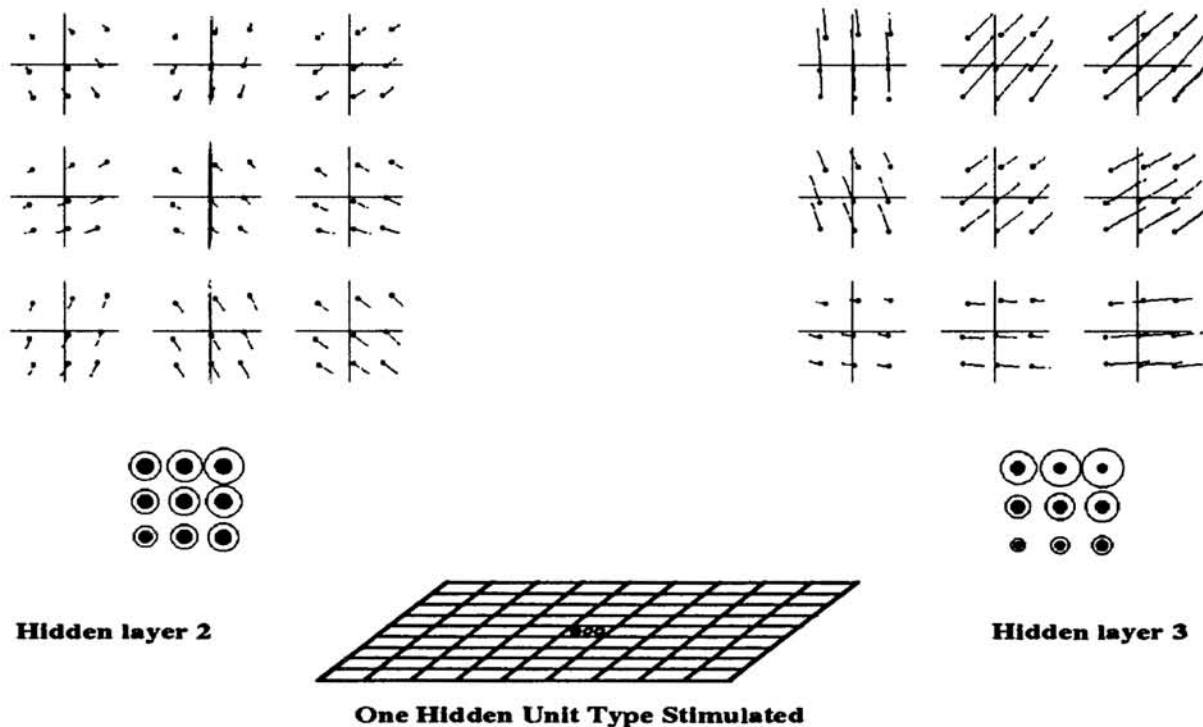

Figure 5: Eye Movements Evoked by Stimulating one Hidden Unit Type

left corner of the map produces an output in the upper left direction, regardless of initial eye position.

There were several types of hidden units at each spatial position of a hidden layer. When the hidden units were stimulated independently, the pattern of induced eye movements was no longer a function solely of the location of the stimulation (figure 5). Other factors, such as the preferred head-centered direction of the stimulated cell, were also important. Hence, the intermediate maps were not purely retinotopic.

If all the hidden units present at one location in a hidden layer were activated together, the pattern of outputs resembled the one obtained by stimulating the input layer (figure 6). Even though each hidden unit has a different preferred head-centered direction, when simultaneously activated, these directions balanced out and the dominant factor became the location of the stimulation.

Strong electrical stimulation in area V1 of the visual cortex is likely to recruit many neurons whose receptive fields share the same retinal location. This might explain why McIlwain [9] observed eye movements in directions that depended only on the position of the stimulation site. In higher visual areas with weaker retinotopy, it might be possible to obtain patterns closer to those produced by stimulating only one type of hidden unit. Such patterns of eye movements have already been observed in parietal area LIP [4].

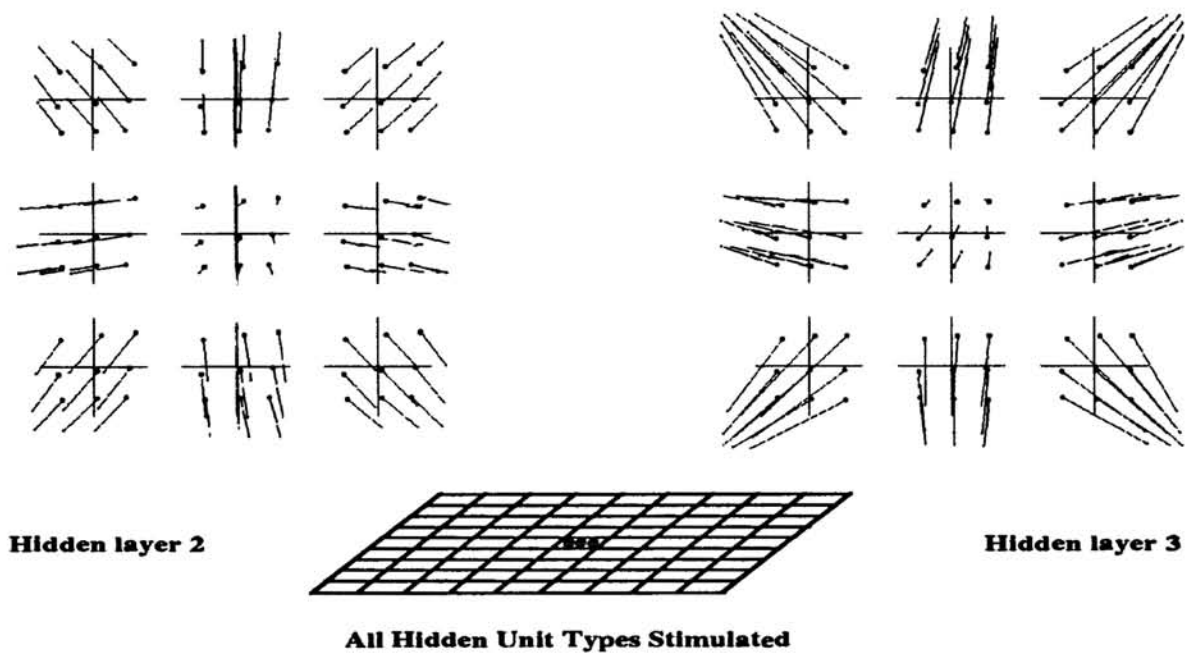

**Hidden layer 2**                **Hidden layer 3**

**All Hidden Unit Types Stimulated**

Figure 6: Eye Movements Evoked by Stimulating all Hidden Unit Types

## 4   Discussion and Predictions

The analysis of our hierarchical model shows that the gain modulation of visual responses observed at early stages of visual processing are consistent with the hypothesis that low-level visual features are encoded in head-centered coordinates. What experiments could confirm this hypothesis?

Electrical stimulation cannot distinguish between a retinotopic and a retinospatiotopic representation unless the number of neurons stimulated is small or restricted to those with similar gain fields. This might be possible in an intermediate level of processing, such as area V3a.

Most psychophysical experiments have been designed to test for purely head-centered maps [10, 5] and not for retinotopic maps receiving a static eye position signal. New experiments are needed that look for interactions between eye position and visual features. For example, it should be possible to obtain motion aftereffects that are dependent on eye position; that is, an aftereffect in which the direction of motion depends on the gaze direction. John Mayhew [8] has already reported this type of gaze-dependent aftereffect for rotation, which is probably represented at later stages of visual processing. Similar experiments with translational motion could probe earlier levels of visual processing.

If information on spatial location is already present in area V1, the primary visual area that projects to other areas of the visual cortex in primates, then we need to re-evaluate the representation of objects in visual cortex. In the model presented here, the spatial location of an object was encoded along with its other features in a distributed fashion; hence spatial location should be considered on equal footing with other features of an object. Such early spatial transformations would affect

other aspects of visual processing, such as visual attention and object recognition, and may also be important for nonspatial tasks, such as shape constancy (John Mayhew, personal communication).

# References

[1] R.A. Andersen, G.K. Essick, and R.M. Siegel. Encoding of spatial location by posterior parietal neurons. *Science*, 230:456–458, 1985.

[2] P.R. Brotchie and R.A. Andersen. A body-centered coordinate system in posterior parietal cortex. In *Neurosc. Abst.*, page 1281, New Orleans, 1991.

[3] C. Galleti and P.P. Battaglini. Gaze-dependent visual neurons in area v3a of monkey prestriate cortex. *J. Neurosc.*, 9:1112–1125, 1989.

[4] S.J. Goodman and R.A. Andersen. Microstimulations of a neural network model for visually guided saccades. *J. Cog. Neurosc.*, 1:317–326, 1989.

[5] D.E. Irwin, J.L. Zacks, and J.S. Brown. Visual memory and the perception of a stable visual environment. *Perc. Psychophy.*, 47:35–46, 1990.

[6] R. Lal and M.J. Freedlander. Gating of retinal transmission by afferent eye position and movement signals. *Science*, 243:93–96, 1989.

[7] Y. LeCun, B. Boser, J.S. Denker, D. Henderson, R.E. Howard, and L.D. Jackel. Backpropagation applied to handwritten zip code recognition. *Neural Computation*, 1:540–566, 1990.

[8] J.E.W. Mayhew. After-effects of movement contingent on direction of gaze. *Vision Res.*, 13:877–880, 1973.

[9] J.T. Mc Ilwain. Saccadic eye movements evoked by electrical stimulation of the cat visual cortex. *Visual Neurosc.*, 1:135–143, 1988.

[10] J.K. O'Regan and A. Levy-Schoen. Integrating visual information from successive fixations : does trans-saccadic fusion exist? *Vision Res.*, 23:765–768, 1983.

[11] D.E. Rumelhart, G.E. Hinton, and R.J. Williams. Learning internal representations by error propagation. In D. E. Rumelhart, J. L. McClelland, and the PDP Research Group, editors, *Parallel Distributed Processing*, volume 1, chapter 8, pages 318–362. MIT Press, Cambridge, MA, 1986.

[12] Y. Trotter, S. Celebrini, S.J. Thorpe, and Imbert M. Modulation of stereoscopic processing in primate visual cortex v1 by the distance of fixation. In *Neurosc. Abs.*, New-Orleans, 1991.

[13] T.G. Weyand and J.G. Malpeli. Responses of neurons in primary visual cortex are influenced by eye position. In *Neurosc. Abs.*, page 419.7, St Louis, 1990.

[14] D. Zipser and R.A. Andersen. A back-propagation programmed network that stimulates reponse properties of a subset of posterior parietal neurons. *Nature*, 331:679–684, 1988.